# Optimization on a Budget: A Reinforcement Learning Approach

**Paul Ruvolo**
Department of Computer Science
University of California San Diego
La Jolla, CA 92093
pruvolo@cs.ucsd.edu

**Ian Fasel**
Department of Computer Sciences
University of Texas at Austin
ianfasel@cs.utexas.edu

**Javier Movellan**
Machine Perception Laboratory
University of California San Diego
movellan@mplab.ucsd.edu

## Abstract

Many popular optimization algorithms, like the Levenberg-Marquardt algorithm (LMA), use heuristic-based "controllers" that modulate the behavior of the optimizer during the optimization process. For example, in the LMA a damping parameter $\lambda$ is dynamically modified based on a set of rules that were developed using heuristic arguments. Reinforcement learning (RL) is a machine learning approach to learn optimal controllers from examples and thus is an obvious candidate to improve the heuristic-based controllers implicit in the most popular and heavily used optimization algorithms.

Improving the performance of off-the-shelf optimizers is particularly important for time-constrained optimization problems. For example the LMA algorithm has become popular for many real-time computer vision problems, including object tracking from video, where only a small amount of time can be allocated to the optimizer on each incoming video frame.

Here we show that a popular modern reinforcement learning technique using a very simple state space can dramatically improve the performance of general purpose optimizers, like the LMA. Surprisingly the controllers learned for a particular domain also work well in very different optimization domains. For example we used RL methods to train a new controller for the damping parameter of the LMA. This controller was trained on a collection of classic, relatively small, non-linear regression problems. The modified LMA performed better than the standard LMA on these problems. This controller also dramatically outperformed the standard LMA on a difficult computer vision problem for which it had not been trained. Thus the controller appeared to have extracted control rules that were not just domain specific but generalized across a range of optimization domains.

## 1 Introduction

Most popular optimization algorithms, like the Levenberg-Marquardt algorithm (LMA) use simple "controllers" that modulate the behavior of the optimization algorithm based on the state of the optimization process. For example, in the LMA a damping factor $\lambda$ modifies the descent step to behave more like Gradient Descent or more like the Gauss-Newton optimization algorithm [1, 2].

The LMA uses the following heuristic for controlling $\lambda$: If an iteration of the LMA with the current damping factor $\lambda_t$ reduces the error then the new parameters produced by the LMA iteration are accepted and the damping factor is divided by a constant term $\eta > 0$, i.e., $\lambda_{t+1} = \lambda_t/\eta$. Otherwise, if the error is not reduced, the new parameters are not accepted, the damping factor is multiplied by $\eta$, and the LMA iteration is repeated with the new damping parameter. While various heuristic arguments have been used to justify this particular way of controlling the damping factor, it is not clear whether this "controller" is optimal in any way or whether it can be significantly improved.

Improving the performance of off-the-shelf optimizers is particularly important for time-constrained optimization problems. For example the LMA algorithm has become popular for many real-time computer vision problems, including object tracking from video, where only a small amount of time can be allocated to the optimizer on each incoming video frame. Time constrained optimization is in fact becoming an increasingly important problem in applications such as operations research, robotics, and machine perception. In these problems the focus is on achieving the best possible solution in a fixed amount of time. Given the special properties of time constrained optimization problems it is likely that the heuristic-based controllers used in off-the-shelf optimizers may not be particularly efficient. Additionally, standard techniques for non-linear optimization like the LMA do not address issues such as when to stop a fruitless local search or when to revisit a previously visited part of the parameter space.

Reinforcement learning (RL) is a machine learning approach to learn optimal controllers by examples and thus is an obvious candidate to improve the heuristic-based controllers used in the most popular and heavily used optimization algorithms. An advantage of RL methods over other approaches to optimal control is that they do not require prior knowledge of the underlying system dynamics and the system designer is free to choose reward metrics that best match the desiderata for controller performance. For example, in the case of optimization under time constraints a suitable reward could be to achieve the minimum loss within a fixed amount of time.

## 2   Related Work

The idea of using RL in optimization problems is not new [3, 4, 5, 6, 7]. However, previous approaches have focused on using RL methods to develop problem-specific optimizers for NP-complete problems. Here our focus is on using RL methods to modify the controllers implicit in the most popular and heavily used optimization algorithms. In particular our goal is to make these algorithms more efficient for optimization on time budget problems. As we will soon show, a simple RL approach can result in dramatic improvements in performance of these popular optimization packages.

There has also been some work on empirical evaluations of the LMA algorithm versus other nonlinear optimization methods in the computer vision community. In [8], the LMA and Powell's dog-leg method are compared on the problem of bundle adjustment. The approach outlined in this document could in principle learn to combine these two methods to perform efficient optimization.

## 3   The Levenberg Marquardt Algorithm

Consider the problem of optimizing a loss function $f : \mathbb{R}^n \to \mathbb{R}$ over the space $\mathbb{R}^n$. There are many approaches to this problem, including zeroth-order methods (such as the Metropolis-Hastings algorithm), first order approaches, such as gradient descent and the Gauss-Newton method, and second order approaches such as the Newton-Raphson algorithm.

Each of these algorithms have advantages and disadvantages. For example, on each iteration of gradient descent, parameters are changed in the opposite direction of the gradient of the loss function, e.g.,

$$x_{k+1} \;\;=\;\; x_k - \eta \bigtriangledown_x f(x_k) \tag{1}$$

Steepest Descent has convergence guarantees provided the value of $\eta$ is reduced over the course of the optimization and in general is robust, but quite slow.

The Gauss-Newton method is a technique for minimizing sums of squares of non-linear functions. Let $g$ be a function from $\mathbb{R}^n \to \mathbb{R}^m$ with a corresponding loss function $L(x) = g(x)^\top g(x)$. The

**if** $f(x_k)^\top f(x_k) > f(x_{k-1})^\top f(x_{k-1})$ **then**
    $x_k \leftarrow x_{k-1}$
    $\lambda \leftarrow \eta \times \lambda$
**else**
    $\lambda \leftarrow \frac{1}{\eta} \times \lambda$
**end if**

Figure 1: A heuristic algorithm for updating lambda during Levenberg-Marquardt non-linear least squares optimization.

algorithm works by first linearizing the function $g$ using its first order Taylor expansion. The sum of squares loss function, $L$, then becomes a quadratic function that can be analytically minimized. Let $H = J(x_k)^\top J(x_k)$ and $d = J(x_k)^\top g(x_k)$, where $J$ is the Jacobian of $g$ with respect to $x$. Each iteration of the Gauss-Newton method is of the following form:

$$x_{k+1} \quad = \quad x_k - H^{-1}d \tag{2}$$

The Gauss-Newton method has a much faster convergence rate than gradient descent, however, it is not as robust as gradient descent. It can actually perform very poorly when the linear approximation to $g$ is not accurate.

Levenberg-Marquardt [1] is a popular optimization algorithm that attempts to blend gradient descent and Gauss-Newton in order to obtain both the fast convergence rate of Gauss-Newton and the convergence guarantees of gradient descent. The algorithm has the following update rule:

$$x_{k+1} \quad = \quad x_k - (H + \lambda diag(H))^{-1}d \tag{3}$$

This update rule is also known as damped Gauss-Newton because the $\lambda$ parameter serves to dampen the Gauss-Newton step by blending it with the gradient descent step. Marquardt proposed a heuristic based control law to dynamically modify $\lambda$ during the optimization process (see Figure 1). This control has become part of most LMA packages.

The LMA algorithm has recently become a very popular approach to solve real-time problems in computer vision [9, 10, 11], such as object tracking and feature tracking in video. Due to the special nature of this problem it is unclear whether the heuristic-based controller embedded in the algorithm is optimal or could be significantly improved upon.

In the remainder of this document we explore whether reinforcement learning methods can help improve the performance of LMA by developing an empirically learned controller of the damping factor rather than the commonly used heuristic controller.

## 4   Learning Control Policies for Optimization Algorithms

An optimizer is an algorithm that uses some statistics about the current progress of the optimization in order to produce a next iterate to evaluate. It is natural to frame optimization in the language of control theory by thinking of the statistics of the optimization progress used by the controller to choose the next iterate as the control state and the next iterate to visit as the control action. In this work we choose to restrict our state space to a few statistics that capture both the current time constraints and the recent progress of the optimization procedure. The action space is restricted by making the observation that current methods for non-linear optimization provide good suggestions for the next point to visit. In this way our action space encodes which one of a fixed set of optimization subroutines (see Section 3) to use for the next iteration, along with actions that control various heuristic parameters for each optimization subroutine (for instance schedules for updating $\eta$ in gradient descent and heuristics for modifying the value of $\lambda$ in the LMA).

In order to define the optimality of a controller we define a reward function that indicates the desirability of the solution found during optimization. In the context of optimization with semi-rigid time constraints an appropriate reward function balances reduction in loss of the objective function with the number of steps needed to achieve that reduction. In the case of optimization with a fixed budget, a more natural choice might be the overall reduction in the loss function within the alloted budget of function evaluations. For specific applications, in a similar spirit to the work of Boyan [6],

Initialize a policy $\pi_0$ that explores randomly
$S \leftarrow \{\}$
**for** $i = 1$ to $n$ **do**
    Generate a random optimization problem $U$
    Optimize $U$ for $T$ time steps using policy $\pi_0$ and generate samples $V \in (s, a, r, s')^T$
    $S \leftarrow S \cup V$
**end for**
**repeat**
    Construct the approximate action-value function $Q_t^{\pi_t}$ using the samples $S$
    Set $\pi_{t+1}$ to be the one step policy improvement of $\pi_t$ using $Q_t^{\pi_t}$
    $t \leftarrow t + 1$
**until** $Q_{t-1}^{\pi} \approx Q_t^{\pi}$
**return** $\pi_t$

Figure 2: Our algorithm for learning controllers for optimization on a budget. The construction of the approximate action-value function and the policy improvement step are performed using the techniques outlined in [12].

the reward function could be modified to include features of intermediate solutions that are likely to indicate the desirability of the current point.

Given a state space, action space, and reward function for a given optimization problem, reinforcement learning methods provide an appropriate set of techniques for learning an optimal optimization controller. While there are many reinforcement learning algorithms that are appropriate for our problem formulation, in this work we employ Least-Squares Policy Iteration (LSPI) [12]. Least Squares Policy Iteration is particularly attractive since it handles continuous state spaces, is efficient in terms of the number of interactions with the system needed to learn a good controller, does not need an underlying model of the process dynamics, and learns models that are amenable to interpretation.

LSPI is an iterative procedure that repeatedly applies the following two steps until convergence: approximating the action-value function as a linear combination of a fixed set of basis functions and then improving the current policy greedily over the approximate value function. The bases are functions of the state and action and can be non-linear. The method is efficient in terms of the number of interactions required with the dynamical system and can reuse the same set of samples to evaluate multiple policies, which is a crucial difference between LSPI and earlier methods like LSTD. The output of the LSPI procedure is a weight vector that defines the action-value function of the optimal policy as a linear combination of the basis vectors.

Our method for learning an optimization controller consists of two phases. In the first phase samples are collected through interactions between a random optimization controller and an optimization problem in a series of fixed length optimization episodes. These samples are tuples of the form $(s, a, r, s')$ where $s'$ denotes the state arrived at when action $a$ was executed starting from state $s$ and reward $r$ was received. The second phase of our algorithm applies LSPI to learn an action-value function and implicitly an optimal policy (which is given by the greedy maximization of the action-value function over actions for a given state). A sketch of our algorithm is given in Figure 2.

## 5 Experiments

We demonstrate the ability of our method to both achieve superior performance to off the shelf non-linear optimization techniques as well as provide insight into the specific policies and action-value functions learned.

### 5.1 Optimizing Nonlinear Least-Squares Functions with a Fixed Budget

Both the classical non-linear problems and the facial expression recognition task were formulated in terms of optimization given a fixed budget of function evaluations. This criterion suggests a natural reward function where $L$ is a loss function we are trying to minimize, $B$ is the budget of function evaluations, $I$ is the indicator function, $x_0$ is the initial point visited in the optimization, and $x_{opt}$ is

the point with the lowest loss visited in the current optimization episode:

$$r_k \quad = \quad I(k < B) \times I(L(x_k) < L(x_{opt})) \times (L(x_{opt}) - L(x_k)) \times \frac{1}{L(x_0)} \qquad (4)$$

This reward function encourages controllers that achieve large reductions in loss within the fixed budget of function evaluations.

Each optimization problem takes the form of minimizing the sum of squares of non-linear functions and thus are well-suited to Levenberg-Marquardt style optimization. The action space we consider in our experiments consists of adjustments to the damping factor (maintain, decrease by a multiplicative factor, or increase by a multiplicative factor) used in the LMA, the decision of whether or not to throw away the last descent step, along with two actions that are not available to the LMA. These additional actions include moving to a new random point in the domain of the objective function and also returning to the best point found so far and performing one descent step using the LMA (using the current damping factor). The number of actions available at each step is 8 (6 for various combinations of adjustments to $\lambda$ and returning the the previous iterate along with the 2 additional actions just described).

The state space used to make the action decision includes a fixed-length window of history that encodes whether a particular step in the past increased or decreased the residual error from the previous iterate. This window is set to size 2 for most of our experiments, however, we did evaluate the relative improvement of using a window size of 1 versus 2 (see Figure 4). Also included in the state space is the amount of function evaluations left in our budget and a problem-specific state feature described in Section 5.3.

The state and action space are mapped through a collection of fixed basis functions which the LSPI algorithm combines linearly to approximate the optimal action-value function. For most applications of LSPI these functions consist of radial-basis functions distributed throughout the continuous state and action space. The basis we use in our problem treats each action independently and thus constructs a tuple of basis functions for each action. To encode the number of evaluations left in the optimization episode, we use a collection of radial-basis functions centered at different values of budget remaining (specifically we use basis functions spaced at 4 step intervals with a bandwidth of .3). The history window of whether the loss went up or down during recent iterations of the algorithm is represented as a $d$-dimensional binary vector where $d$ is the length of history window considered. For the facial expression recognition task the tuple includes an additional basis described in Section 5.3.

## 5.2   Classical Nonlinear Least Squares Problems

In order to validate our approach we apply it to a dataset of classical non-linear optimization problems [13]. This dataset of problems includes famous optimization problems that cover a wide variety of non-linear behavior. Examples include the Kowalik and Osborne function and the Scaled Meyer function. When restricted to a budget of 5 function evaluations, our method is able to learn a policy which results in a 6% gain in performance (measured in total reduction in loss from the starting point) when compared to the LMA.

## 5.3   Learning to Classify Facial Expressions

The box-filter features that proved successful for face detection in [14] have also shown promise for recognizing facial expressions when combined using boosting methods. The response of a box-filter to an image patch is obtained by weighting the sum of the pixel brightnesses in various boxes by a coefficient defined by the particular box-filter kernel. In our work we frame the problem of feature selection as an optimization procedure over a continuous parameter space. The parameter space defines an infinite set of box-filters that includes many of those proposed in [14] as a special case (see Figure 3). Each feature can be described as a vector in $[0, 1]^6$ where the 6 dimensions of the vector are depicted in Figure 3.

We learn a detector for the presence or absence of a smile using the pixel intensities of an image patch containing a face. We accomplish this by employing the sequential regression procedure L2-boost [15]. L2-boost creates a strong classifier by iteratively fitting the residuals of the current model

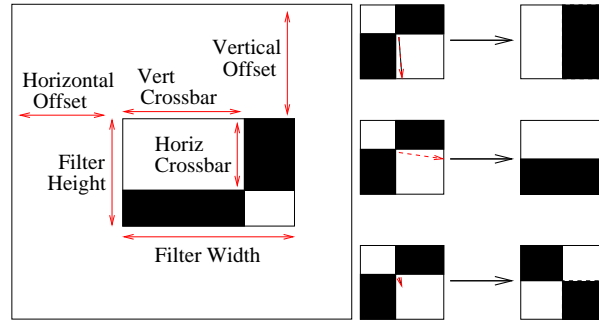

Figure 3: A parameterized feature space. The position of the cross-hairs in the middle of the box filter can freely float. This added generality allows for the features proposed in [14] to be generated as special cases. A complete description of a feature is composed of the 6 parameters depicted above: horizontal offset, vertical crossbar, vertical offset, filter height, horizontal crossbar, and filter width. The weighting coefficients for the four boxes (depicted in a checkerboard pattern) is determined by linear regression between filter outputs of each box and the labels of the training set.

over a collection of weak-learners (in this case our parameterized features). The L2-boost procedure selects a box-filter at each iteration that most reduces the difference between the current predictions of the model and the correct image labels. Once a sufficiently good feature is found this feature is added to the current ensemble. L2-boost learns a linear model for predicting the label of the image patch since each weak learner (box-filter) is a linear filter on the pixel values and L2-boost combines weak learners in a linear fashion. The basis space for LSPI is augmented for this task by included a basis that specifies the number of features already selected by the L2-boost procedure.

We test our algorithm on the task of smile detection using a subset of $1,000$ images from the GENKI dataset (which is a collection of $60,000$ faces from the web). Along with information about the location of faces and facial features, human labelers have labeled each image as containing or not containing a smile. In this experiment our goal is to predict the human smile labels using the L2-boost procedure outlined above.

During each trial 3 box filters are selected using the L2-boosting procedure. Within each round of feature selection a total of 20 feature evaluations are allowed per round. We use the default version of the LMA as a mode of comparison. After collecting samples from 100 episodes of optimization on the GENKI dataset, LSPI is able to learn a policy that achieves a 2.66 fold greater reduction in total loss than the LMA on a test set of faces from the GENKI dataset (see Figure 4). Since the LMA does not have access to the ability to move to a new random part of the state space a more fair comparison would be to our method without access to this action. In this experiment our method is still able to achieve a $20\%$ greater reduction in total loss than the LMA.

Figure 4 shows that the policies learned using our method not only achieves greater reduction in loss on the training set, but that this reduction in loss translates to a significant gain in performance for classification on a validation set of test images. Our method achieves between .036 and .083 better classification performance (as measured by area under the ROC curve) depending on the optimization budget. Note that given the relatively high baseline performance of the LMA on the smile detection task, an improvement of .083 in terms of area under the ROC translates to almost halving the error rate. Also of significance is that the information encoded in the state space does make a difference in the performance of the algorithm. Learning a policy that uses a history window of error changes on the last two time steps is able to achieve a $16\%$ greater reduction in total loss than a policy learned with a history window of size 1.

Also of interest is the nature of the policies learned for smile detection on a fixed budget. The policies learned exhibit the following general trend: during the early stages of selecting a specific feature the learned policies either sample a new point in the feature space (if the error has increased from the last iteration) or do a Levenberg-Marquardt step on the best point visited up until now (if the error has gone down at the last iteration). This initial strategy makes sense since if the current point does not look promising (error has increased) it is wise to try a different part of the state space,

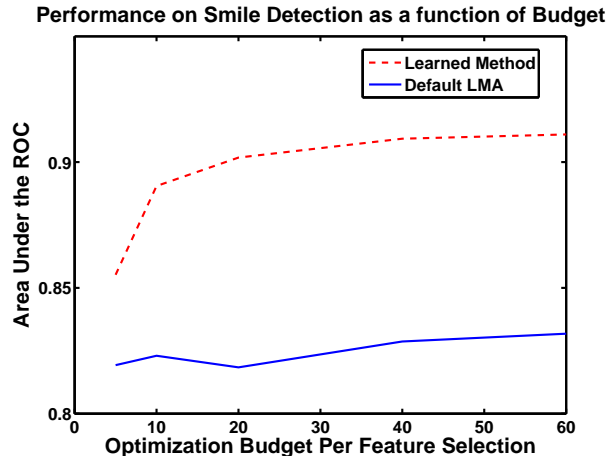

Performance on Smile Detection as a function of Budget

| Controller Type | Average Reduction in Loss Relative to the LMA |
|---|---|
| *Learned (history window = 1)* | 2.3 |
| *Learned (history window = 2)* | 2.66 |
| *Learned (no random restarts)* | 1.2 |
| *Learned on Classical (no random restarts)* | 1.19 |
| *Default LMA* | 1.0 |

Figure 4: **Top**: The performance on detecting smile versus not smile is substantially better when using an optimization controller learned with our algorithm than using the default LMA. In each run 3 features are selected by the L2-boost procedure. The number of feature evaluations per feature (the budget) varies along the x-axis. **Bottom:** This table describes the relative improvement in total loss reduction for policies learned using our method.

however, if the error is decreasing it is best to continue to apply local optimization methods. Later in the optimization, the policy always performs a Levenberg-Marquardt step on the current best point no matter what the change in error was. This strategy makes sense since once a few different parts of the state space have been investigated the utility of sampling a new part of the state space is reduced. Several trends can be seen by examining the basis weights learned by LSPI. The first trend is that the learned policy favors discarding the last iterate versus keeping (similar to the LMA). The second trend is that the policy favors increasing the damping parameter when the error has increased on the last iteration and decreasing the damping factor when the error has decreased (also similar to the LMA).

## 5.4 Cross Generalization

A property of choosing a general state space for our method is that the policies learned on one class of optimization problem are applicable to other classes of optimization. The optimization controllers learned in the classical least squares minimization task achieve a 19% improvement over the standard LMA on the smile detection task. Applying the controllers learned on the smile detection task to the classical least squares problem yields a more modest 5% improvement. These results support the claim that our method is extracting useful structure for optimizing under a fixed budget and not simply learning a controller that is amenable to a particular problem domain.

## 6 Conclusion

We have presented a novel approach to the problem of learning optimization procedures for optimization on a fixed budget. We have shown that our approach achieves better performance than ubiquitous methods for non-linear least squares optimization on the task of optimizing within a fixed budget of function evaluations for both classical non-linear functions and a difficult computer vision task. We have also provided an analysis of the patterns learned by our method and how they

make sense in the context of optimization under a fixed budget. Additionally, we have presented extensions to the features used in [14] that are significant in their own right.

In the future we will more fully explore the framework that we have outlined in this document. The specific application of the framework in the current work (state, action, and bases) while quite effective may be able to be improved. For instance, by incorporating domain specific features into the state space richer policies might be learned. We also want to apply this technique to other problems in machine perception. An upcoming project will test the viability of our technique for finding feature point locations on a face that simultaneously exhibit high likelihood in terms of appearance and high likelihood in terms of the relative arrangement of facial features. The real-time constraints of this problem make it a particularly appropriate target for the methods presented in this document.

# References

[1] K. Levenberg, "A method for the solution of certain problems in least squares," *Applied Math Quarterly*, 1944.

[2] D. Marquardt, "An algorithm for least-squares estimation of nonlinear parameters," *SIAM Journal of Applied Mathematics*, 1963.

[3] V. V. Miagkikh and W. F. P. III, "Global search in combinatorial optimization using reinforcement learning algorithms," in *Proceedings of the Congress on Evolutionary Computation*, vol. 1. IEEE Press, 6-9 1999, pp. 189–196.

[4] Y. Zhang, "Solving large-scale linear programs by interior-point methods under the MATLAB environment," *Optimization Methods and Software*, vol. 10, pp. 1–31, 1998.

[5] L. M. Gambardella and M. Dorigo, "Ant-q: A reinforcement learning approach to the traveling salesman problem," in *International Conference on Machine Learning*, 1995, pp. 252–260.

[6] J. A. Boyan and A. W. Moore, "Learning evaluation functions for global optimization and boolean satisfiability," in *AAAI/IAAI*, 1998, pp. 3–10.

[7] R. Moll, T. J. Perkins, and A. G. Barto, "Machine learning for subproblem selection," in *ICML '00: Proceedings of the Seventeenth International Conference on Machine Learning*. San Francisco, CA, USA: Morgan Kaufmann Publishers Inc., 2000, pp. 615–622.

[8] M. I. Lourakis and A. A. Argyros, "Is levenberg-marquardt the most efficient optimization algorithm for implementing bundle adjustment?" *Proceedings of ICCV*, 2005.

[9] D. Cristinacce and T. F. Cootes, "Feature detection and tracking with constrained local models," *BMVC*, pp. 929–938, 2006.

[10] M. Pollefeys, L. V. Gool, M. Vergauwen, F. Verbiest, K. Cornelis, J. Tops, and R. Koch, "Visual modeling with a hand-held camera," *IJCV*, vol. 59, no. 3, pp. 207–232, 2004.

[11] P. Beardsley, P. Torr, and A. Zisserman, "3d model acquisition from extended image sequences." *Proceedings of ECCV*, pp. 683–695, 1996.

[12] M. Lagoudakis and R. Parr, "Least-squares policy iteration," *Journal of Machine Learning Research*, 2003.

[13] H. B. Nielsen, "Uctp problems for unconstrained optimization," *Technical Report, Technical University of Denmark*, 2000.

[14] P. Viola and M. Jones, "Robust real-time object detection," *International Journal of Computer Vision*, 2002.

[15] P. Buhlmann and B. Yu, "Boosting with the l2 loss: Regression and classification," *Journal of the American Statistical Association*, 2003.

